# Large Margin DAGs for Multiclass Classification

**John C. Platt**
Microsoft Research
1 Microsoft Way
Redmond, WA 98052
*jplatt@microsoft.com*

**Nello Cristianini**
Dept. of Engineering Mathematics
University of Bristol
Bristol, BS8 1TR - UK
*nello.cristianini@bristol.ac.uk*

**John Shawe-Taylor**
Department of Computer Science
Royal Holloway College - University of London
EGHAM, Surrey, TW20 0EX - UK
*j.shawe-taylor@dcs.rhbnc.ac.uk*

## Abstract

We present a new learning architecture: the Decision Directed Acyclic Graph (DDAG), which is used to combine many two-class classifiers into a multiclass classifier. For an $N$-class problem, the DDAG contains $N(N-1)/2$ classifiers, one for each pair of classes. We present a VC analysis of the case when the node classifiers are hyperplanes; the resulting bound on the test error depends on $N$ and on the margin achieved at the nodes, but not on the dimension of the space. This motivates an algorithm, DAGSVM, which operates in a kernel-induced feature space and uses two-class maximal margin hyperplanes at each decision-node of the DDAG. The DAGSVM is substantially faster to train and evaluate than either the standard algorithm or Max Wins, while maintaining comparable accuracy to both of these algorithms.

## 1 Introduction

The problem of multiclass classification, especially for systems like SVMs, doesn't present an easy solution. It is generally simpler to construct classifier theory and algorithms for two mutually-exclusive classes than for $N$ mutually-exclusive classes. We believe constructing $N$-class SVMs is still an unsolved research problem.

The standard method for $N$-class SVMs [10] is to construct $N$ SVMs. The $i$th SVM will be trained with all of the examples in the $i$th class with positive labels, and all other examples with negative labels. We refer to SVMs trained in this way as *1-v-r* SVMs (short for one-versus-rest). The final output of the $N$ 1-v-r SVMs is the class that corresponds to the SVM with the highest output value. Unfortunately, there is no bound on the generalization error for the 1-v-r SVM, and the training time of the standard method scales linearly with $N$.

Another method for constructing $N$-class classifiers from SVMs is derived from previous research into combining two-class classifiers. Knerr [5] suggested constructing all possible two-class classifiers from a training set of $N$ classes, each classifier being trained on only

two out of $N$ classes. There would thus be $K = N(N-1)/2$ classifiers. When applied to SVMs, we refer to this as *1-v-1* SVMs (short for one-versus-one).

Knerr suggested combining these two-class classifiers with an "AND" gate [5]. Friedman [4] suggested a Max Wins algorithm: each 1-v-1 classifier casts one vote for its preferred class, and the final result is the class with the most votes. Friedman shows circumstances in which this algorithm is Bayes optimal. Kreßel [6] applies the Max Wins algorithm to Support Vector Machines with excellent results.

A significant disadvantage of the 1-v-1 approach, however, is that, unless the individual classifiers are carefully regularized (as in SVMs), the overall $N$-class classifier system will tend to overfit. The "AND" combination method and the Max Wins combination method do not have bounds on the generalization error. Finally, the size of the 1-v-1 classifier may grow superlinearly with $N$, and hence, may be slow to evaluate on large problems.

In Section 2, we introduce a new multiclass learning architecture, called the Decision Directed Acyclic Graph (DDAG). The DDAG contains $N(N-1)/2$ nodes, each with an associated 1-v-1 classifier. In Section 3, we present a VC analysis of DDAGs whose classifiers are hyperplanes, showing that the margins achieved at the decision nodes and the size of the graph both affect their performance, while the dimensionality of the input space does not. The VC analysis indicates that building large margin DAGs in high-dimensional feature spaces can yield good generalization performance. Using such bound as a guide, in Section 4, we introduce a novel algorithm for multiclass classification based on placing 1-v-1 SVMs into nodes of a DDAG. This algorithm, called DAGSVM, is efficient to train and evaluate. Empirical evidence of this efficiency is shown in Section 5.

## 2   Decision DAGs

A Directed Acyclic Graph (DAG) is a graph whose edges have an orientation and no cycles. A Rooted DAG has a unique node such that it is the only node which has no arcs pointing into it. A Rooted Binary DAG has nodes which have either 0 or 2 arcs leaving them. We will use Rooted Binary DAGs in order to define a class of functions to be used in classification tasks. The class of functions computed by Rooted Binary DAGs is formally defined as follows.

**Definition 1** Decision DAGs (DDAGs). *Given a space $X$ and a set of boolean functions $\mathcal{F} = \{f : X \to \{0, 1\}\}$, the class $\mathrm{DDAG}(\mathcal{F})$ of Decision DAGs on $N$ classes over $\mathcal{F}$ are functions which can be implemented using a rooted binary DAG with $N$ leaves labeled by the classes where each of the $K = N(N-1)/2$ internal nodes is labeled with an element of $\mathcal{F}$. The nodes are arranged in a triangle with the single root node at the top, two nodes in the second layer and so on until the final layer of $N$ leaves. The $i$-th node in layer $j < N$ is connected to the $i$-th and $(i + 1)$-st node in the $(j + 1)$-st layer.*

To evaluate a particular DDAG $G$ on input $x \in X$, starting at the root node, the binary function at a node is evaluated. The node is then exited via the left edge, if the binary function is zero; or the right edge, if the binary function is one. The next node's binary function is then evaluated. The value of the decision function $D(x)$ is the value associated with the final leaf node (see Figure 1(a)). The path taken through the DDAG is known as the *evaluation path*. The input $x$ reaches a node of the graph, if that node is on the evaluation path for $x$. We refer to the decision node distinguishing classes $i$ and $j$ as the $ij$-node. Assuming that the number of a leaf is its class, this node is the $i$-th node in the $(N - j + i)$-th layer provided $i < j$. Similarly the $j$-nodes are those nodes involving class $j$, that is, the internal nodes on the two diagonals containing the leaf labeled by $j$.

The DDAG is equivalent to operating on a list, where each node eliminates one class from the list. The list is initialized with a list of all classes. A test point is evaluated against the decision node that corresponds to the first and last elements of the list. If the node prefers

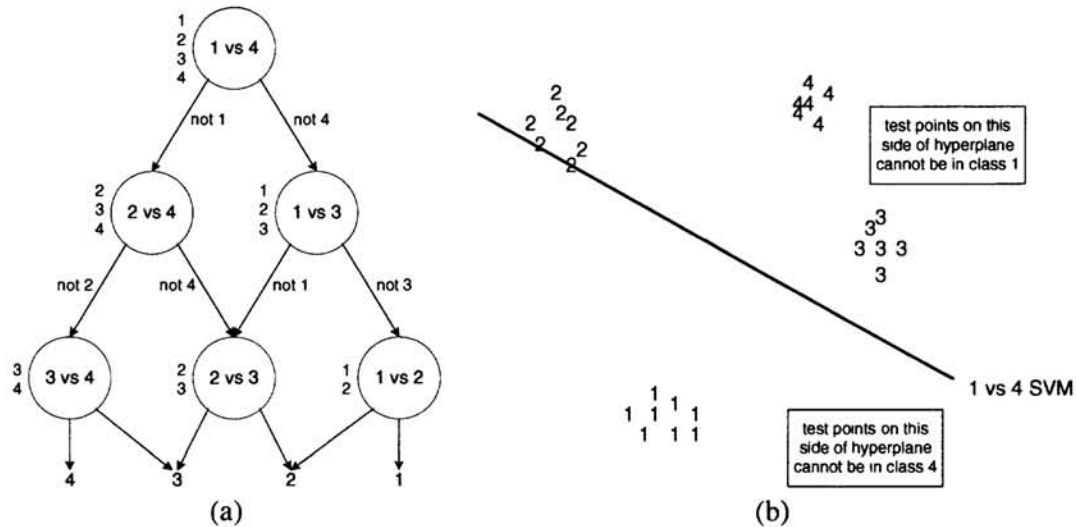

Figure 1: (a) The decision DAG for finding the best class out of four classes. The equivalent list state for each node is shown next to that node. (b) A diagram of the input space of a four-class problem. A 1-v-1 SVM can only exclude one class from consideration.

one of the two classes, the other class is eliminated from the list, and the DDAG proceeds to test the first and last elements of the new list. The DDAG terminates when only one class remains in the list. Thus, for a problem with $N$ classes, $N - 1$ decision nodes will be evaluated in order to derive an answer.

The current state of the list is the total state of the system. Therefore, since a list state is reachable in more than one possible path through the system, the decision graph the algorithm traverses is a DAG, not simply a tree.

Decision DAGs naturally generalize the class of Decision Trees, allowing for a more efficient representation of redundancies and repetitions that can occur in different branches of the tree, by allowing the merging of different decision paths. The class of functions implemented is the same as that of Generalized Decision Trees [1], but this particular representation presents both computational and learning-theoretical advantages.

## 3 Analysis of Generalization

In this paper we study DDAGs where the node-classifiers are hyperplanes. We define a *Perceptron DDAG* to be a DDAG with a perceptron at every node. Let $w$ be the (unit) weight vector correctly splitting the $i$ and $j$ classes at the $ij$-node with threshold $\theta$. We define the margin of the $ij$-node to be $\gamma = \min_{c(x)=i,j} \{|\langle w, x \rangle - \theta|\}$, where $c(x)$ is the class associated to training example $x$. Note that, in this definition, we only take into account examples with class labels equal to $i$ or $j$.

**Theorem 1** *Suppose we are able to classify a random $m$ sample of labeled examples using a Perceptron DDAG on $N$ classes containing $K$ decision nodes with margins $\gamma_i$ at node $i$, then we can bound the generalization error with probability greater than $1 - \delta$ to be less than*

$$\frac{130R^2}{m} \left( D' \log(4em) \log(4m) + \log \frac{2(2m)^K}{\delta} \right),$$

*where $D' = \sum_{i=1}^{K} \frac{1}{\gamma_i^2}$, and $R$ is the radius of a ball containing the distribution's support.*

**Proof**: see Appendix □

Theorem 1 implies that we can control the capacity of DDAGs by enlarging their margin. Note that, in some situations, this bound may be pessimistic: the DDAG partitions the input space into polytopic regions, each of which is mapped to a leaf node and assigned to a specific class. Intuitively, the only margins that should matter are the ones relative to the boundaries of the cell where a given training point is assigned, whereas the bound in Theorem 1 depends on all the margins in the graph.

By the above observations, we would expect that a DDAG whose $j$-node margins are large would be accurate at identifying class $j$, even when other nodes do not have large margins. Theorem 2 substantiates this by showing that the appropriate bound depends only on the $j$-node margins, but first we introduce the notation, $\epsilon_j(G) = P\{x : (x \text{ in class } j \text{ and } x \text{ is misclassified by } G) \text{ or } x \text{ is misclassified as class } j \text{ by } G\}$.

**Theorem 2** *Suppose we are able to correctly distinguish class $j$ from the other classes in a random $m$-sample with a DDAG $G$ over $N$ classes containing $K$ decision nodes with margins $\gamma_i$ at node $i$, then with probability $1 - \delta$,*

$$\epsilon_j(G) \leq \frac{130R^2}{m}\left(D'\log(4em)\log(4m) + \log\frac{2(2m)^{N-1}}{\delta}\right),$$

*where $D' = \sum_{i\in j\text{-nodes}} \frac{1}{\gamma_i^2}$, and $R$ is the radius of a ball containing the support of the distribution.*

**Proof**: follows exactly Lemma 4 and Theorem 1, but is omitted.□

## 4   The DAGSVM algorithm

Based on the previous analysis, we propose a new algorithm, called the Directed Acyclic Graph SVM (DAGSVM) algorithm, which combines the results of 1-v-1 SVMs. We will show that this combination method is efficient to train and evaluate.

The analysis of Section 3 indicates that maximizing the margin of all of the nodes in a DDAG will minimize a bound on the generalization error. This bound is also independent of input dimensionality. Therefore, we will create a DDAG whose nodes are maximum margin classifiers over a kernel-induced feature space. Such a DDAG is obtained by training each $ij$-node only on the subset of training points labeled by $i$ or $j$. The final class decision is derived by using the DDAG architecture, described in Section 2.

The DAGSVM separates the individual classes with large margin. It is safe to discard the losing class at each 1-v-1 decision because, for the hard margin case, all of the examples of the losing class are far away from the decision surface (see Figure 1(b)).

For the DAGSVM, the choice of the class order in the list (or DDAG) is arbitrary. The experiments in Section 5 simply use a list of classes in the natural numerical (or alphabetical) order. Limited experimentation with re-ordering the list did not yield significant changes in accuracy performance.

The DAGSVM algorithm is superior to other multiclass SVM algorithms in both training and evaluation time. Empirically, SVM training is observed to scale super-linearly with the training set size $m$ [7], according to a power law: $T = cm^\gamma$, where $\gamma \approx 2$ for algorithms based on the decomposition method, with some proportionality constant $c$. For the standard 1-v-r multiclass SVM training algorithm, the entire training set is used to create all $N$ classifiers. Hence the training time for 1-v-r is

$$T_{1-v-r} = cNm^\gamma. \tag{1}$$

Assuming that the classes have the same number of examples, training each 1-v-1 SVM only requires $2m/N$ training examples. Thus, training $K$ 1-v-1 SVMs would require

$$T_{1-v-1} = c\frac{N(N-1)}{2}\left(\frac{2m}{N}\right)^\gamma \approx 2^{\gamma-1}cN^{2-\gamma}m^\gamma. \tag{2}$$

For a typical case, where $\gamma = 2$, the amount of time required to train all of the 1-v-1 SVMs is independent of $N$, and is only twice that of training a single 1-v-r SVM. Using 1-v-1 SVMs with a combination algorithm is thus preferred for training time.

## 5 Empirical Comparisons and Conclusions

The DAGSVM algorithm was evaluated on three different test sets: the USPS handwritten digit data set [10], the UCI Letter data set [2], and the UCI Covertype data set [2]. The USPS digit data consists of 10 classes (0-9), whose inputs are pixels of a scaled input image. There are 7291 training examples and 2007 test examples. The UCI Letter data consists of 26 classes (A-Z), whose inputs are measured statistics of printed font glyphs. We used the first 16000 examples for training, and the last 4000 for testing. All inputs of the UCI Letter data set were scaled to lie in $[-1, 1]$. The UCI Covertype data consists of 7 classes of trees, where the inputs are terrain features. There are 11340 training examples and 565893 test examples. All of the continuous inputs for Covertype were scaled to have zero mean and unit variance. Discrete inputs were represented as a 1-of-n code.

On each data set, we trained $N$ 1-v-r SVMs and $K$ 1-v-1 SVMs, using SMO [7], with soft margins. We combined the 1-v-1 SVMs both with the Max Wins algorithm and with DAGSVM. The choice of kernel and of the regularizing parameter $C$ was determined via performance on a validation set. The validation performance was measured by training on 70% of the training set and testing the combination algorithm on 30% of the training set (except for Covertype, where the UCI validation set was used). The best kernel was selected from a set of polynomial kernels (from degree 1 through 6), both homogeneous and inhomogeneous; and Gaussian kernels, with various $\sigma$. The Gaussian kernel was always found to be best.

| | $\sigma$ | $C$ | Error Rate (%) | Kernel Evaluations | Training CPU Time (sec) | Classifier Size (Kparameters) |
|---|---|---|---|---|---|---|
| **USPS** | | | | | | |
| 1-v-r | 3.58 | 100 | 4.7 | 2936 | 3532 | 760 |
| Max Wins | 5.06 | 100 | 4.5 | 1877 | 307 | 487 |
| DAGSVM | 5.06 | 100 | 4.4 | 819 | 307 | 487 |
| Neural Net [10] | | | 5.9 | | | |
| **UCI Letter** | | | | | | |
| 1-v-r | 0.447 | 100 | 2.2 | 8183 | 1764 | 148 |
| Max Wins | 0.632 | 100 | 2.4 | 7357 | 441 | 160 |
| DAGSVM | 0.447 | 10 | 2.2 | 3834 | 792 | 223 |
| Neural Net | | | 4.3 | | | |
| **UCI Covertype** | | | | | | |
| 1-v-r | 1 | 10 | 30.2 | 7366 | 4210 | 105 |
| Max Wins | 1 | 10 | **29.0** | 7238 | 1305 | 107 |
| DAGSVM | 1 | 10 | 29.2 | 4390 | 1305 | 107 |
| Neural Net [2] | | | 30 | | | |

Table 1: Experimental Results

Table 1 shows the results of the experiments. The optimal parameters for all three multiclass SVM algorithms are very similar for both data sets. Also, the error rates are similar for all three algorithms for both data sets. Neither 1-v-r nor Max Wins is statistically significantly better than DAGSVM using McNemar's test [3] at a 0.05 significance level for USPS or UCI Letter. For UCI Covertype, Max Wins is slightly better than either of the other SVM-based algorithms. The results for a neural network trained on the same data sets are shown for a baseline accuracy comparison.

The three algorithms distinguish themselves in training time, evaluation time, and classifier size. The number of kernel evaluations is a good indication of evaluation time. For 1-v-

r and Max Wins, the number of kernel evaluations is the total number of unique support vectors for all SVMs. For the DAGSVM, the number of kernel evaluations is the number of unique support vectors averaged over the evaluation paths through the DDAG taken by the test set. As can be seen in Table 1, Max Wins is faster than 1-v-r SVMs, due to shared support vectors between the 1-v-1 classifiers. The DAGSVM has the fastest evaluation. The DAGSVM is between a factor of 1.6 and 2.3 times faster to evaluate than Max Wins.

The DAGSVM algorithm is also substantially faster to train than the standard 1-v-r SVM algorithm: a factor of 2.2 and 11.5 times faster for these two data sets. The Max Wins algorithm shares a similar training speed advantage.

Because the SVM basis functions are drawn from a limited set, they can be shared across classifiers for a great savings in classifier size. The number of parameters for DAGSVM (and Max Wins) is comparable to the number of parameters for 1-v-r SVM, even though there are $N(N-1)/2$ classifiers, rather than $N$.

In summary, we have created a Decision DAG architecture, which is amenable to a VC-style bound of generalization error. Using this bound, we created the DAGSVM algorithm, which places a two-class SVM at every node of the DDAG. The DAGSVM algorithm was tested versus the standard 1-v-r multiclass SVM algorithm, and Friedman's Max Wins combination algorithm. The DAGSVM algorithm yields comparable accuracy and memory usage to the other two algorithms, but yields substantial improvements in both training and evaluation time.

## 6 Appendix: Proof of Main Theorem

**Definition 2** *Let $\mathcal{F}$ be a set of real valued functions. We say that a set of points $X$ is $\gamma$-shattered by $\mathcal{F}$ relative to $r = (r_x)_{x \in X}$, if there are real numbers $r_x$, indexed by $x \in X$, such that for all binary vectors $b$ indexed by $X$, there is a function $f_b \in \mathcal{F}$ satisfying $(2b_x - 1)f_b(x) \geq (2b_x - 1)r_x + \gamma$. The fat shattering dimension, $\mathrm{fat}_{\mathcal{F}}$, of the set $\mathcal{F}$ is a function from the positive real numbers to the integers which maps a value $\gamma$ to the size of the largest $\gamma$-shattered set, if the set is finite, or maps to infinity otherwise.*

As a relevant example, consider the class $\mathcal{F}_{\mathrm{lin}} = \{x \to \langle w, x \rangle - \theta : \|w\| = 1\}$. We quote the following result from [1].

**Theorem 3** *Let $\mathcal{F}_{\mathrm{lin}}$ be restricted to points in a ball of $n$ dimensions of radius $R$ about the origin. Then*

$$\mathrm{fat}_{\mathcal{F}_{\mathrm{lin}}}(\gamma) \leq \min\{R^2/\gamma^2, n+1\}.$$

We will bound generalization with a technique that closely resembles the technique used in [1] to study Perceptron Decision Trees. We will now give a lemma and a theorem: the lemma bounds the probability over a double sample that the first half has zero error and the second error greater than an appropriate $\epsilon$. We assume that the DDAG on $N$ classes has $K = N(N-1)/2$ nodes and we denote $\mathrm{fat}_{\mathcal{F}_{\mathrm{lin}}}(\gamma)$ by $\mathrm{fat}(\gamma)$.

**Lemma 4** *Let $G$ be a DDAG on $N$ classes with $K = N(N-1)/2$ decision nodes with margins $\gamma^1, \gamma^2, \ldots, \gamma^K$ at the decision nodes satisfying $k_i = \mathrm{fat}(\gamma_i/8)$, where $\mathrm{fat}$ is continuous from the right. Then the following bound holds, $P^{2m}\{\mathbf{xy} : \exists$ a graph $G : G$ which separates classes $i$ and $j$ at the $ij$-node for all $x$ in $\mathbf{x}$, a fraction of points misclassified in $y > \epsilon(m, K, \delta).\} < \delta$ where $\epsilon(m, K, \delta) = \frac{1}{m}(D \log(8m) + \log \frac{2^K}{\delta})$ and $D = \sum_{i=1}^{K} k_i \log(4em/k_i)$.*

**Proof** The proof of Lemma 4 is omitted for space reasons, but is formally analogous to the proof of Lemma 4.4 in [8], and can easily be reconstructed from it. □

Lemma 4 applies to a particular DDAG with a specified margin $\gamma_i$ at each node. In practice, we observe these quantities after generating the DDAG. Hence, to obtain a bound that can be applied in practice, we must bound the probabilities uniformly over all of the possible margins that can arise. We can now give the proof for Theorem 1.

**Proof of Main Theorem**: We must bound the probabilities over different margins. We first use a standard result due to Vapnik [9, page 168] to bound the probability of error in terms of the probability of the discrepancy between the performance on two halves of a double sample. Then we combine this result with Lemma 4. We must consider all possible patterns of $k_i$'s over the decision nodes. The largest allowed value of $k_i$ is $m$, and so, for fixed $K$, we can bound the number of possibilities by $m^K$. Hence, there are $m^K$ of applications of Lemma 4 for a fixed $N$. Since $K = N(N-1)/2$, we can let $\delta_k = \delta/m^K$, so that the sum $\sum_{k=1}^{m} \delta_k = \delta$. Choosing

$$\epsilon\left(m, K, \frac{\delta_k}{2}\right) = \frac{65R^2}{m}\left(D'\log(4em)\log(4m) + \log\frac{2(2m)^K}{\delta}\right) \tag{3}$$

in the applications of Lemma 4 ensures that the probability of any of the statements failing to hold is less than $\delta/2$. Note that we have replaced the constant $8^2 = 64$ by 65 in order to ensure the continuity from the right required for the application of Lemma 4 and have upper bounded $\log(4em/k_i)$ by $\log(4em)$. Applying Vapnik's Lemma [9, page 168] in each case, the probability that the statement of the theorem fails to hold is less than $\delta$. □

More details on this style of proof, omitted in this paper for space constraints, can be found in [1].

# References

[1] K. Bennett, N. Cristianini, J. Shawe-Taylor, and D. Wu. Enlarging the margin in perceptron decision trees. *Machine Learning (submitted)*. http://lara.enm.bris.ac.uk/cig/pubs/ML-PDT.ps.

[2] C. Blake, E. Keogh, and C. Merz. UCI repository of machine learning databases. Dept. of information and computer sciences, University of California, Irvine, 1998. http://www.ics.uci.edu/~mlearn/MLRepository.html.

[3] T. G. Dietterich. Approximate statistical tests for comparing supervised classification learning algorithms. *Neural Computation*, 10:1895–1924, 1998.

[4] J. H. Friedman. Another approach to polychotomous classification. Technical report, Stanford Department of Statistics, 1996. http://www-stat.stanford.edu/reports/friedman/poly.ps.Z.

[5] S. Knerr, L. Personnaz, and G. Dreyfus. Single-layer learning revisited: A stepwise procedure for building and training a neural network. In Fogelman-Soulie and Herault, editors, *Neurocomputing: Algorithms, Architectures and Applications*, NATO ASI. Springer, 1990.

[6] U. Kreßel. Pairwise classification and support vector machines. In B. Schölkopf, C. J. C. Burges, and A. J. Smola, editors, *Advances in Kernel Methods: Support Vector Learning*, pages 255–268. MIT Press, Cambridge, MA, 1999.

[7] J. Platt. Fast training of support vector machines using sequential minimal optimization. In B. Schölkopf, C. J. C. Burges, and A. J. Smola, editors, *Advances in Kernel Methods — Support Vector Learning*, pages 185–208. MIT Press, Cambridge, MA, 1999.

[8] J. Shawe-Taylor and N. Cristianini. Data dependent structural risk minimization for perceptron decision trees. In M. Jordan, M. Kearns, and S. Solla, editors, *Advances in Neural Information Processing Systems*, volume 10, pages 336–342. MIT Press, 1999.

[9] V. Vapnik. *Estimation of Dependences Based on Empirical Data [in Russian]*. Nauka, Moscow, 1979. (English translation: Springer Verlag, New York, 1982).

[10] V. Vapnik. *Statistical Learning Theory*. Wiley, New York, 1998.
